# Covariance Kernels from Bayesian Generative Models

**Matthias Seeger**
Institute for Adaptive and Neural Computation
University of Edinburgh
5 Forrest Hill, Edinburgh EH1 2QL
*seeger@dai.ed.ac.uk*

## Abstract

We propose the framework of mutual information kernels for learning covariance kernels, as used in Support Vector machines and Gaussian process classifiers, from unlabeled task data using Bayesian techniques. We describe an implementation of this framework which uses variational Bayesian mixtures of factor analyzers in order to attack classification problems in high-dimensional spaces where labeled data is sparse, but unlabeled data is abundant.

## 1 Introduction

Kernel machines, such as *Support Vector machines* or *Gaussian processes*, are powerful and frequently used tools for solving statistical learning problems. They are based on the use of a *kernel function* which encodes task prior knowledge in a Bayesian manner. In this paper, we propose the framework of *mutual information (MI) kernels* for learning covariance kernels from unlabeled task data using Bayesian techniques. This section introduces terms and concepts. We also discuss some general ideas for discriminative semi-supervised learning and kernel design in this context. In section 2, we define the general framework and give examples. We note that the *Fisher kernel* [4] is a special case of a MI kernel. MI kernels for mixture models are discussed in detail. In section 3, we describe an implementation for a MI kernel for *variational Bayesian mixtures of factor analyzers* models and show results of preliminary experiments.

In the *semi-supervised classification problem*, a labeled dataset $D_l = \{(\boldsymbol{x}_1, t_1), \ldots, (\boldsymbol{x}_m, t_m)\}$ as well as an unlabeled set $D_u = \{\boldsymbol{x}_{m+1}, \ldots, \boldsymbol{x}_{m+n}\}$ are given for training, both i.i.d. drawn from the same unknown distribution, but the labels for $D_u$ cannot be observed. Here, $\boldsymbol{x}_i \in \mathbb{R}^p$ and $t_i \in \{-1, +1\}$.[1] Typically, $m = |D_l|$ is rather small, and $n = |D_u| \gg m$. Our aim is to fit models to $D_u$ in a Bayesian way, thereby extracting (posterior) information, then use this information to build a covariance kernel $K$. Afterwards, $K$ will be plugged into a supervised kernel machine, which is trained on the labeled data $D_l$ to perform the classification task.

It is important to distinguish very clearly between these two learning scenarios. For fitting $D_u$, we use *Bayesian* density estimation. After having chosen a model family $\{P(\boldsymbol{x}|\boldsymbol{\theta})\}$ and a *prior distribution* $P(\boldsymbol{\theta})$ over parameters $\boldsymbol{\theta}$, the *posterior distribution* $P(\boldsymbol{\theta}|D_u) \propto P(D_u|\boldsymbol{\theta})P(\boldsymbol{\theta})$, where $P(D_u|\boldsymbol{\theta}) = \prod_{i=m+1}^{m+n} P(\boldsymbol{x}_i|\boldsymbol{\theta})$, encodes all information that $D_u$ contains about the *latent* (i.e. unobserved) parameters $\boldsymbol{\theta}$.[2] The other learning scenario is supervised classification, using a kernel machine. Such architectures model a smooth *latent function* $y(\boldsymbol{x}) \in \mathbb{R}$ as a *random process*, together with a classification noise model $P(t|y)$.[3] The covariance kernel $K$ specifies the prior distribution for this process: namely, a-priori, $y(\boldsymbol{x})$ is assumed to be a *Gaussian process* with zero mean and covariance function $K$, i.e. $K(\boldsymbol{x}^{(1)}, \boldsymbol{x}^{(2)}) = E[y(\boldsymbol{x}^{(1)})y(\boldsymbol{x}^{(2)})]$; see e.g. [10] for details. In the following, we use the notation $\boldsymbol{a} = (a_i)_i = (a_1, \ldots, a_I)'$ for vectors, and $\mathcal{A} = (a_{i,j})_{i,j}$ for matrices respectively. The prime denotes transposition. $\operatorname{diag} \boldsymbol{a}$ is the matrix with diagonal $\boldsymbol{a}$ and 0 elsewhere. $N(\boldsymbol{x}|\boldsymbol{\mu}, \Sigma)$ denotes the Gaussian density with mean $\boldsymbol{\mu}$ and covariance matrix $\Sigma$.

Within the standard discriminative Bayesian classification scenario, unlabeled data cannot be used. However, it is rather straightforward to modify this scenario by introducing the concept of *conditional priors* (see [6]). If we have a discriminant model family $\{P(t|\boldsymbol{x}; \boldsymbol{w})\}$, a conditional prior $P(\boldsymbol{w}|\boldsymbol{\theta})$ allows to encode prior knowledge and assumptions about how information about $P(\boldsymbol{x})$ (i.e. about $\boldsymbol{\theta}$) influences our assumptions about a-priori probabilities over discriminants $\boldsymbol{w}$. For example, the $P(\boldsymbol{w}|\boldsymbol{\theta})$ could be Occam priors, expressing the intuitive fact that for many problems, the notion of "simplicity" of a discriminant function depends strongly on what is known about the input distribution $P(\boldsymbol{x})$. For a given problem, it is in general not easy to come up with a useful conditional prior. However, once such a prior is specified, we can in principle use the same powerful techniques for approximate Bayesian inference that have been developed for supervised discriminative settings. Semi-supervised techniques that can be seen as employing conditional priors include *co-training* [1], feature selection based on clustering [7] and the Fisher kernel [4]. For a probabilistic kernel technique, $P(\boldsymbol{w}|\boldsymbol{\theta})$ is fully specified by a covariance function $K(\boldsymbol{x}^{(1)}, \boldsymbol{x}^{(2)}|\boldsymbol{\theta})$ depending on $\boldsymbol{\theta}$. The problem is therefore to find covariance kernels which (as GP priors) favour discriminants in some sense compatible with what we have learned about the input distribution $P(\boldsymbol{x})$.

Kernel techniques can be seen as nonparametric smoothers, based on the (prior) assumption that if two input points are "similar" (e.g. "close" under some distance), their labels (and latent outputs $y$) should be highly correlated. Thus, one generic way of learning kernels from unlabeled data is to learn a distance between input points from the information about $P(\boldsymbol{x})$. A frequently used assumption about how classification labels may depend on $P(\boldsymbol{x})$ is the *cluster hypothesis*: we assume discriminants whose decision boundaries lie between clusters in $P(\boldsymbol{x})$ to be a-priori more likely than such that label clusters inconsistently. A general way of encoding this hypothesis is to learn a distance from $P(\boldsymbol{x})$ which is consistent with clusters in $P(\boldsymbol{x})$, i.e. points within the same cluster are closer under this distance than points from different clusters. We can then try to embed the learned distance $d(\boldsymbol{x}^{(1)}, \boldsymbol{x}^{(2)})$ approximately in an Euclidean space, i.e. learn a mapping $\phi : \boldsymbol{x} \mapsto \phi(\boldsymbol{x}) \in \mathbb{R}^l$ such that $d(\boldsymbol{x}^{(1)}, \boldsymbol{x}^{(2)}) \approx \|\phi(\boldsymbol{x}^{(1)}) - \phi(\boldsymbol{x}^{(2)})\|$ for all pairs from $D_u$. Then, a natural kernel function would be $K(\boldsymbol{x}^{(1)}, \boldsymbol{x}^{(2)}) = \exp(-\beta\|\phi(\boldsymbol{x}^{(1)}) - \phi(\boldsymbol{x}^{(2)})\|^2)$. In this paper, however, we follow a simpler approach, by considering a similarity measure

which immediately gives rise to a covariance kernel, without having to compute an approximate Euclidean embedding.

*Remark*: Our main aim in this paper is to construct kernels that can be learned from *unlabeled data only*. In contrast to this, the task of learning a kernel from labeled data is somewhat simpler and can be approached in the following generic way: start with a parametric model family $\{y(\boldsymbol{x}; \boldsymbol{w})\}$, with the interpretation that $y(\boldsymbol{x}; \boldsymbol{w})$ models the log odds $\log(P(t=+1|\boldsymbol{x})/P(t=-1|\boldsymbol{x}))$. Fitting these models to labeled data $D_l$, we obtain a posterior $P(\boldsymbol{w}|D_l)$. Now, a natural covariance kernel for our problem is simply $K(\boldsymbol{x}^{(1)}, \boldsymbol{x}^{(2)}) = \int y(\boldsymbol{x}^{(1)}; \boldsymbol{w})y(\boldsymbol{x}^{(2)}; \boldsymbol{w})Q(\boldsymbol{w})\, d\boldsymbol{w}$, where (say) $Q(\boldsymbol{w}) \propto P(\boldsymbol{w}|D_l)^\lambda P(\boldsymbol{w})^{1-\lambda}$ (or an approximation thereof). For $\lambda = 0$, we obtain the prior covariance kernel for our model, while for larger $\lambda$ the kernel incorporates more and more posterior information. The kernel proposed in [8] can be seen as approximation to this approach.

## 2  Mutual Information Kernels

In this section, we begin by introducing the framework of *mutual information kernels*. Given a *mediator distribution* $P_{med}(\boldsymbol{\theta})$ over parameters $\boldsymbol{\theta}$, we define the joint distribution $Q(\boldsymbol{x}^{(1)}, \boldsymbol{x}^{(2)})$ *mediated by* $P_{med}(\boldsymbol{\theta})$ as

$$Q(\boldsymbol{x}^{(1)}, \boldsymbol{x}^{(2)}) = \int P_{med}(\boldsymbol{\theta})P(\boldsymbol{x}^{(1)}|\boldsymbol{\theta})P(\boldsymbol{x}^{(2)}|\boldsymbol{\theta})\, d\boldsymbol{\theta}. \tag{1}$$

The sample mutual information between $\boldsymbol{x}^{(1)}$ and $\boldsymbol{x}^{(2)}$ under this distribution is

$$I(\boldsymbol{x}^{(1)}, \boldsymbol{x}^{(2)}) = \log \frac{Q(\boldsymbol{x}^{(1)}, \boldsymbol{x}^{(2)})}{Q(\boldsymbol{x}^{(1)})Q(\boldsymbol{x}^{(2)})}, \tag{2}$$

where $Q(\boldsymbol{x}) = \int Q(\boldsymbol{x}, \tilde{\boldsymbol{x}})\, d\tilde{\boldsymbol{x}}$. $I(\boldsymbol{x}^{(1)}, \boldsymbol{x}^{(2)})$ is called the *mutual information (MI) score*. In a very concrete sense, it measures the *similarity* between $\boldsymbol{x}^{(1)}$ and $\boldsymbol{x}^{(2)}$ *with respect to the generative process represented by the mediator distribution* $P_{med}(\boldsymbol{\theta})$: it is the amount of information they *share* via the mediator variable $\boldsymbol{\theta} \sim P_{med}(\boldsymbol{\theta})$. Note that $Q(\boldsymbol{x}^{(1)}, \boldsymbol{x}^{(2)})$ can be seen as *inner product* in a space of functions $\boldsymbol{\theta} \mapsto \mathbb{R}$, the *features* of $\boldsymbol{x}^{(k)}$ being $(P(\boldsymbol{x}^{(k)}|\boldsymbol{\theta}))_{\boldsymbol{\theta}}$, *weighted* by the distribution $P_{med}$.[4] $\boldsymbol{x}^{(k)}$ is represented by its likelihood under *all* possible models.

Covariance kernels have to satisfy the property of *positive definiteness*[5], and the MI score $I$ does not. However, applying a standard transformation (called *exponential embedding (EE)* here), we arrive at

$$K(\boldsymbol{x}^{(1)}, \boldsymbol{x}^{(2)}) = e^{-(I(\mathbf{x}^{(1)}, \mathbf{x}^{(1)}) + I(\mathbf{x}^{(2)}, \mathbf{x}^{(2)}))/2 + I(\mathbf{x}^{(1)}, \mathbf{x}^{(2)})} = \frac{Q(\boldsymbol{x}^{(1)}, \boldsymbol{x}^{(2)})}{\sqrt{Q(\boldsymbol{x}^{(1)}, \boldsymbol{x}^{(1)})Q(\boldsymbol{x}^{(2)}, \boldsymbol{x}^{(2)})}}. \tag{3}$$

EE becomes familiar if we note that it transforms the standard inner product $\boldsymbol{x}^{(1)'}\boldsymbol{x}^{(2)}$ into the well-known *Radial Basis Function (RBF) kernel*[6]

$$K_{RBF}(\boldsymbol{x}^{(1)}, \boldsymbol{x}^{(2)}) = e^{-\frac{\beta}{2}\|\mathbf{x}^{(1)} - \mathbf{x}^{(2)}\|^2}, \tag{4}$$

or the weighted inner product $\boldsymbol{x}^{(1)'}\mathcal{D}\boldsymbol{x}^{(2)}$ into the *squared-exponential* kernel (e.g. [10]). It is easy to show that $K$ in (3) is a valid covariance kernel[7], and we refer to it as *mutual information (MI) kernel*.

Example: Let $P(\boldsymbol{x}|\boldsymbol{\theta}) = N(\boldsymbol{x}|\boldsymbol{\theta}, (\rho/2)\mathcal{I})$ (spherical Gaussian with mean $\boldsymbol{\theta}$), $P_{med}(\boldsymbol{\theta}) = N(\boldsymbol{\theta}|\tilde{\boldsymbol{\theta}}, \alpha\mathcal{I})$. Then, the MI kernel $K$ is the RBF kernel (4) with $\beta = 4/(\rho(4 + \rho/\alpha))$. Thus, the RBF kernel is a special case of a MI kernel.

## 2.1 Mediator distribution. Model-trust scaling.

The mediator distribution $P_{med}(\boldsymbol{\theta})$, motivated earlier in this section, should ideally encode information about the $\boldsymbol{x}$ generation process, just as the Bayesian posterior $P(\boldsymbol{\theta}|D_u)$. On the other hand, we need to be able to control the influence that information from sources such as unlabeled data $D_u$ can have on the kernel (relying too much on such sources results in lack of robustness, see e.g. [6] for details). Here, we propose *model-trust scaling (MTS)*, by setting

$$P_{med}(\boldsymbol{\theta}|\lambda) \propto P(D_u|\boldsymbol{\theta})^{\lambda/n} P(\boldsymbol{\theta}), \quad \lambda \in [0, n]. \tag{5}$$

$P_{med}$ varies with $\lambda$ from the (usually vague) prior $P(\boldsymbol{\theta})$ ($\lambda = 0$) towards the sharp posterior $P(\boldsymbol{\theta}|D_u)$ ($\lambda = n$), rendering the $D_u$ information (via the model) more and more influence upon the kernel $K$. The concrete effect of MTS on the kernel depends on the model family.

Example (continued): Again, $P(\boldsymbol{x}|\boldsymbol{\theta}) = N(\boldsymbol{x}|\boldsymbol{\theta}, (\rho/2)\mathcal{I})$, with a flat prior $P(\boldsymbol{\theta}) = 1$ on the mean. Then, $P(\boldsymbol{\theta}|D_u) = N(\boldsymbol{\theta}|\bar{\boldsymbol{x}}, (\rho/2n)\mathcal{I})$, where $\bar{\boldsymbol{x}} = n^{-1}\sum_{i=m+1}^{m+n} \boldsymbol{x}_i$, and $P_{med}(\boldsymbol{\theta}) = N(\boldsymbol{\theta}|\bar{\boldsymbol{x}}, (\rho/2\lambda)\mathcal{I})$ (after (5)). Thus, the MI kernel is again the RBF kernel (4) with $\beta = 2/(\rho(2 + \lambda))$. For the more flexible model $P(\boldsymbol{x}|\boldsymbol{\theta}) = N(\boldsymbol{x}|\boldsymbol{\mu}, \Sigma)$, $\boldsymbol{\theta} = (\boldsymbol{\mu}, \Sigma)$ and the conjugate Jeffreys prior, the MI kernel is computed in [5].

If the Bayesian analysis is done with conjugate prior-model pairs, the corresponding MI kernel can be computed easily, and for many of these cases, MTS has a very simple, analytic form (see [5]). In general, approximation techniques developed for Bayesian analysis have to be applied. For example, applying the *Laplace approximation* to the computations on a model with flat prior $P(\boldsymbol{\theta}) = 1$ results in the *Fisher kernel* [4][8], see e.g. [5]. However, in this paper we favour another approximation technique (see section 3).

## 2.2 Mutual Information Kernels for Mixture Models

If we apply the MI kernel framework to *mixture models* $P(\boldsymbol{x}|\boldsymbol{\theta}, \boldsymbol{\pi}) = \sum_s \pi_s P(\boldsymbol{x}|\boldsymbol{\theta}_s)$, we run into a problem. As mentioned in section 1, we would like our kernel at least partly to encode the *cluster hypothesis*, i.e. $K(\boldsymbol{x}^{(1)}, \boldsymbol{x}^{(2)})$ should be small if $\boldsymbol{x}^{(1)}, \boldsymbol{x}^{(2)}$ come from different clusters in $P(\boldsymbol{x})$,[9] but the opposite is true (for not too small

$\lambda$). To overcome this problem, we generalize $Q(\boldsymbol{x}^{(1)}, \boldsymbol{x}^{(2)})$:

$$Q(\boldsymbol{x}^{(1)}, \boldsymbol{x}^{(2)}) = \sum_{s_1,s_2=1}^{S} w_{s_1,s_2} \int P(\boldsymbol{x}^{(1)}|\boldsymbol{\theta}_{s_1}) P(\boldsymbol{x}^{(2)}|\boldsymbol{\theta}_{s_2}) P_{med}(\boldsymbol{\theta}) \, d\boldsymbol{\theta}, \qquad (6)$$

where $\mathcal{W} = (w_{s_1,s_2})_{s_1,s_2}$ is symmetric with nonnegative entries and positive elements on the diagonal. The MI kernel $K$ is defined as before by (3), based on the new $Q$. If $P_{med}(\boldsymbol{\theta}, \boldsymbol{\pi}) = \prod_s P_{med}(\boldsymbol{\theta}_s) P_{med}(\boldsymbol{\pi})$ (which is true for the cases we will be interested in), we see that the original MI kernel arises as special case $w_{s_1,s_2} = E_{P_{med}}[\pi_{s_1}\pi_{s_2}]$. Now, by choosing $\mathcal{W} = \mathrm{diag}(E_{P_{med}}[\pi_s])_s$, we arrive at a MI kernel $K$ which (typically) behaves as expected w.r.t. cluster separation (see figure 1), but does not exhibit long-range correlations between joined components. In the present work, we restrict ourselves to this *diagonal mixture kernel*. Note that this kernel can be seen as (normalized) mixture of MI kernels over the component models.

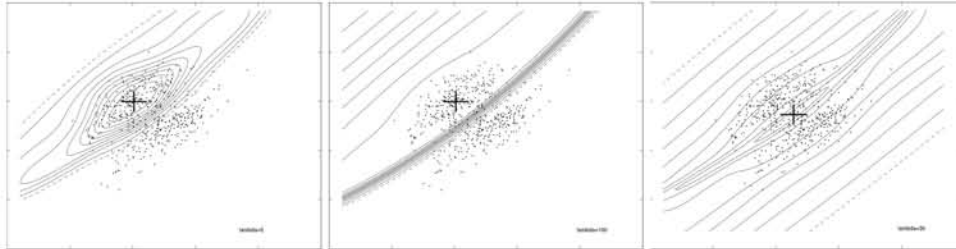

Figure 1: Kernel contours on 2-cluster dataset ($\lambda = 5, 100, 30$)

Figure 1 shows contour plots[10] of the diagonal mixture kernel for VB-MoFA (see section 3), learned on a 500 cases dataset sampled from two Gaussians with equal covariance (see subsection 3.1). We plot $K(\boldsymbol{a}, \boldsymbol{x})$ for fixed $\boldsymbol{a}$ (marked by a cross) against all $\boldsymbol{x}$, the height between contour lines is 0.1. The left and middle plot have the lower cluster's centre as $\boldsymbol{a}$, with $\lambda = 5$, $\lambda = 100$ respectively, the right plot's $\boldsymbol{a}$ lies between the cluster centres, $\lambda = 30$. The effect of MTS can be seen by comparing left and middle, note the different sharpness of the slopes towards the other cluster and the different sizes and shapes of the "high correlation" regions. As seen on the right, points *between* clusters have highest correlation with other such inter-cluster points, a feature that may be very useful for successful discrimination.

## 3  Experiments with Mixtures of Factor Analyzers

In this section, we describe an implementation of a MI kernel, using *variational Bayesian mixtures of factor analyzers (VB-MoFA)* [2] as density models. These are able to combine local dimensionality reduction (using noisy linear transformations $\boldsymbol{u} \to \boldsymbol{x}$ from low-dimensional latent spaces) with good global data fit using mixtures. VB-MoFA is a *variational approximation* to Bayesian analysis on these models, able to deliver the posterior approximations we require for an MI kernel.

We employ the diagonal mixture kernel (see subsection 2.2). Instead of implementing MTS analytically, we compute the VB approximation to the true posterior (i.e. $\lambda = n$), then simply apply the scaling to this distribution. $P_{med}(\boldsymbol{\theta}, \boldsymbol{\pi})$ factorizes as required in subsection 2.2. The integrals $\int P(\boldsymbol{x}^{(1)}|\boldsymbol{\theta}_s) P(\boldsymbol{x}^{(2)}|\boldsymbol{\theta}_s) P_{med}(\boldsymbol{\theta}_s) \, d\boldsymbol{\theta}_s$ in (6)

are not analytically tractable. Our first idea was to approximate them by applying the VB technique once more, ending up with what we called *one-step variational* approximations. Unfortunately, the MI kernel approximation based on these terms cannot be shown to be positive definite anymore[11]! Thus, in the moment we use a less elegant and, we feel, less accurate approximation (details can be found in [5]) based on first-order Taylor expansions.

In the remainder of this section we compare the VB-MoFA kernel with the RBF kernel (4) on two datasets, using a Laplace GP classifier (see [10]). In each case we sample a training pool, a kernel dataset $D_u$ and a test set (mutually exclusive). The VB-MoFA diagonal mixture kernel is learned on $D_u$. For a given training set size $m$, a *run* consists of sampling a training set $D_l$ and a holdout set $D_h$ (both of size $m$) from the training pool, tuning kernel parameters by validation on $D_h$, then testing on the test set. We use the same $D_l, D_u$ for both kernels. For each training set size, we do $L = 30$ runs. Results are presented by plotting means and 95% $t$-test confidence intervals of test errors over runs.

## 3.1 Two Gaussian clusters

The dataset is sampled from two 2-d Gaussians with same non-spherical covariance (see figure 1), one for each class (the Bayes error is 2.64%). We use $n = 500$ points for $D_u$, a training pool of 100 and a test set of 500 points. The learning curves in figure 2 show that on this simple toy problem, on which the fitted VB-MoFA model represents the cluster structure in $P(\boldsymbol{x})$ almost perfectly, the VB-MoFA MI kernel outperforms the RBF kernel for samples sizes $n \leq 40$.

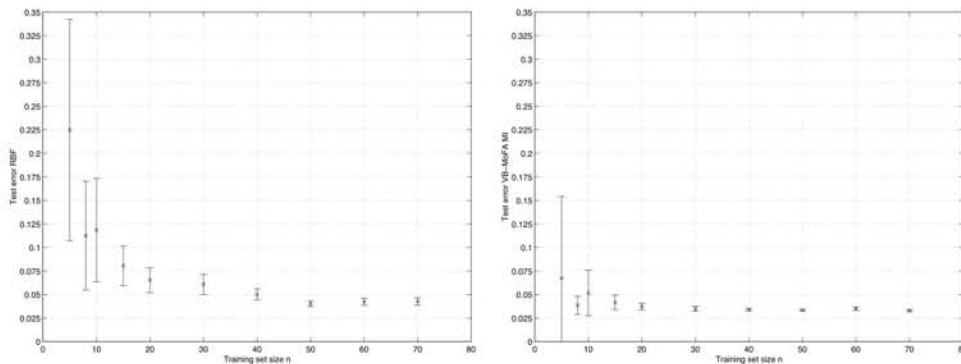

Figure 2: Learning curves on 2-cluster dataset. Left: RBF kernel; right: MI kernel

## 3.2 Handwritten Digits (MNIST): Twos against threes

We report results of preliminary experiments using the subset of twos and threes of the MNIST Handwritten Digits database[12]. Here, $n = |D_u| = 2000$, the training pool contains 8089, the test set 2000 cases. We employ a VB-MoFA model with 20 components, fitted to $D_u$. We use a very simple baseline (BL) algorithm (see [6], section 2.3) based on the component densities from the VB-MoFA model[13], which

allows us to assess the "purity" of the component clusters w.r.t. the labels[14]; this algorithm is the only one *not* based on a kernel. Furthermore, we show results for the one-step variational approximation to the MI kernel[15] (MIOLD). The learning curves are shown in figure 3.

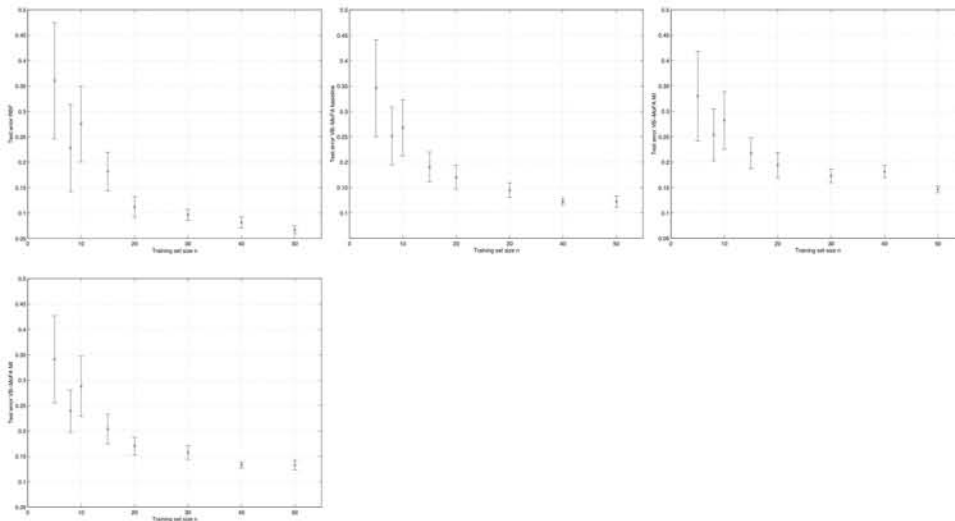

Figure 3: Learning curves on MNIST twos/threes. Upper left: RBF kernel; upper middle: Baseline method; upper right: VB-MoFA MI kernel (first-order approx.); lower left: VB-MoFA MI "kernel" (one-step var. approx.)

The results are disappointing. The fact that the first-order approximation to the MI kernel performs worse than the one-step variational approximation (although the latter may fail to be positive definite), indicates that the former is a poorer approximation. The latter renders results close to the baseline method, while the smoothing RBF kernel makes much better use of a growing number of labeled examples[16] This indicates that the conditional prior, as represented by the VB-MoFA MI kernel, behaves nonsmooth and overrides label information in regions where it should not. We suspect this problem to be related to the high dimensionality of the input space, in which case probability densities tend to have a large dynamic range, and mixture component responsibility estimates tend to behave very nonsmooth. Thus, it seems to be necessary to extend the basic MI kernel framework by new scaling mechanisms in order to produce a smoother encoding of the prior assumptions.

# 4 Related work. Discussion

The present work is probably most closely related to the *Fisher kernel* (see subsection 2.1). The arguments concerning mixture models (see subsection 2.2) apply there as well. Haussler [3] contains a wealth of material about kernel design for discrete objects $x$. Watkins [9] mentions that expressions like $Q$ in (1) are valid kernels for discrete $x$ and countable parameter spaces. Very recently we came across [11], which essentially describes a special case of the diagonal mixture kernel (see subsection 2.2) for Gaussian components with diagonal covariances[17]. The author calls $Q$ a *stochastic equivalence predicate*. He is interested in distance learning, does not apply his method to kernel machines and does not give a Bayesian interpretation.

We have presented a general framework for kernel learning from unlabeled data and described an approximate implementation using VB-MoFA models. A straightforward application of this technique to high-dimensional real-world data did not prove successful, and in future work we will explore new ideas for extending the basic MI kernel framework in order to be able to deal with high-dimensional input spaces.

## Acknowledgments

We thank Chris Williams for many inspiring discussions, furthermore Ralf Herbrich, Amos Storkey, Hugo Zaragoza and Neil Lawrence. Matt Beal helped us a lot with VB-MoFA. The author gratefully acknowledges support through a research studentship from *Microsoft Research Ltd.*

## Footnotes

[1] For simplicity, we only discuss binary labels here.

[2]In practice, computation of $P(\boldsymbol{\theta}|D_u)$ is hardly ever feasible, but powerful approximation techniques can be used.

[3]A natural choice for binary classification is to represent the log odds $\log(P(t = +1|\boldsymbol{x})/P(t = -1|\boldsymbol{x}))$ by $y(\boldsymbol{x})$.

[4] When comparing $\boldsymbol{x}^{(1)}, \boldsymbol{x}^{(2)}$ via the inner product, we are not interested in correlating their features uniformly, but rather focus on regions of high volume under $P_{med}$.

[5] $K$ is *positive definite* if the matrix $(K(\boldsymbol{x}^{(k_1)}, \boldsymbol{x}^{(k_2)}))_{k_1, k_2}$ is positive definite for *every* set $\{\boldsymbol{x}^{(1)}, \ldots, \boldsymbol{x}^{(K)}\}$ of distinct points.

[6] One can show that if $\hat{I}$ is itself a kernel, and $\hat{I} \to \hat{K}$ under EE, then $\hat{K}^\beta$ is also a kernel for all $\beta > 0$ (see e.g. [3]).

[7]$Q(\boldsymbol{x}^{(1)}, \boldsymbol{x}^{(2)})$ *is* an inner product (therefore a kernel), for the rest of the argument see [3], section 5.

[8]This was essentially observed by the authors of [4] on workshop talks, but has not been published to our knowledge. The fascinating idea of the Fisher kernel has indeed been the main motivation and inspiration for this paper.

[9]This does *not* mean that we (a-priori) believe they should have different labels, but only that the label (or better: the latent $y(\cdot)$) at one of them should not depend strongly on $y(\cdot)$ at the other.

[10]Produced using the first-order approximation (see 3) to the MI kernel. Plots using the one-step variational approximation (see 3) have a somewhat richer structure.

[11]Thanks to an anonymous reviewer for pointing out this flaw.

[12]The $28 \times 28$ images were downsampled to size $8 \times 8$.

[13]The estimates $P(\boldsymbol{x}|s)$ are obtained by integrating out the parameters $\boldsymbol{\theta}_s$ using the variational posterior approximation. The integral is not analytic, and we use a one-step variational approximation to it.

[14]The baseline algorithm is based on the assumption that, given the component index $s$, the input point $x$ and the label $t$ are independent. Only the conditional probabilities $P(t|s)$ are learned, while $P(x|s)$ and $P(s)$ is obtained from the VB-MoFA model fitted to unlabeled data only. Thus, success/failure of this method should be closely related to the degree of purity of the component clusters w.r.t. the labels.

[15]This is somewhat inconsistent, since we use a kernel function which might not be positive definite in a context (GP classification) which requires a covariance function.

[16]Note also that RBF kernel matrices can be evaluated significantly faster than such using the VB-MoFA MI kernel.

[17] The $\alpha$ parameter in this work is related to MTS in this case.

## References

[1] Avrim Blum and Tom Mitchell. Combining labeled and unlabeled data with Co-Training. In *Proceedings of COLT*, 1998.

[2] Z. Ghahramani and M. Beal. Variational inference for Bayesian mixtures of factor analysers. In *Advances in NIPS 12*. MIT Press, 1999.

[3] David Haussler. Convolution kernels on discrete structures. Technical Report UCSC-CRL-99-10, University of California, Santa Cruz, July 1999.

[4] Tommi S. Jaakkola and David Haussler. Exploiting generative models in discriminative classifiers. In *Advances in Neural Information Processing Systems 11*, 1998.

[5] Matthias Seeger. Covariance kernels from Bayesian generative models. Technical report, 2000. Available at `http://www.dai.ed.ac.uk/~seeger/papers.html`.

[6] Matthias Seeger. Learning with labeled and unlabeled data. Technical report, 2000. Available at `http://www.dai.ed.ac.uk/~seeger/papers.html`.

[7] Martin Szummer and Tommi Jaakkola. Partially labeled classification with Markov random walks. In *Advances in NIPS 14*. MIT Press, 2001.

[8] Koji Tsuda, Motoaki Kawanabe, Gunnar Rätsch, Soeren Sonnenburg, and Klaus-Robert Müller. A new discriminative kernel from probabilistic models. In *Advances in NIPS 14*. MIT Press, 2001.

[9] Chris Watkins. Dynamic alignment kernels. Technical Report CSD–TR–98–11, Royal Holloway, University of London, 1999.

[10] Christopher K.I. Williams and David Barber. Bayesian classification with Gaussian processes. *IEEE Trans. PAMI*, 20(12):1342–1351, 1998.

[11] Peter Yianilos. Metric learning via normal mixtures. Technical report, NEC Research, Princeton, 1995.

